# The Fast Convergence of Boosting

**Matus Telgarsky**
Department of Computer Science and Engineering
University of California, San Diego
9500 Gilman Drive, La Jolla, CA 92093-0404
`mtelgars@cs.ucsd.edu`

## Abstract

This manuscript considers the convergence rate of boosting under a large class of losses, including the exponential and logistic losses, where the best previous rate of convergence was $\mathcal{O}(\exp(1/\epsilon^2))$. First, it is established that the setting of weak learnability aids the entire class, granting a rate $\mathcal{O}(\ln(1/\epsilon))$. Next, the (disjoint) conditions under which the infimal empirical risk is attainable are characterized in terms of the sample and weak learning class, and a new proof is given for the known rate $\mathcal{O}(\ln(1/\epsilon))$. Finally, it is established that any instance can be decomposed into two smaller instances resembling the two preceding special cases, yielding a rate $\mathcal{O}(1/\epsilon)$, with a matching lower bound for the logistic loss. The principal technical hurdle throughout this work is the potential unattainability of the infimal empirical risk; the technique for overcoming this barrier may be of general interest.

## 1   Introduction

Boosting is the task of converting inaccurate *weak learners* into a single accurate predictor. The existence of any such method was unknown until the breakthrough result of Schapire [1]: under a *weak learning assumption*, it is possible to combine many carefully chosen weak learners into a majority of majorities with arbitrarily low training error. Soon after, Freund [2] noted that a single majority is enough, and that $\mathcal{O}(\ln(1/\epsilon))$ iterations are both necessary and sufficient to attain accuracy $\epsilon$. Finally, their combined effort produced AdaBoost, which attains the optimal convergence rate (under the weak learning assumption), and has an astonishingly simple implementation [3].

It was eventually revealed that AdaBoost was minimizing a risk functional, specifically the exponential loss [4]. Aiming to alleviate perceived deficiencies in the algorithm, other loss functions were proposed, foremost amongst these being the logistic loss [5]. Given the wide practical success of boosting with the logistic loss, it is perhaps surprising that no convergence rate better than $\mathcal{O}(\exp(1/\epsilon^2))$ was known, even under the weak learning assumption [6]. The reason for this deficiency is simple: unlike SVM, least squares, and basically any other optimization problem considered in machine learning, there might not exist a choice which attains the minimal risk! This reliance is carried over from convex optimization, where the assumption of attainability is generally made, either directly, or through stronger conditions like compact level sets or strong convexity [7].

Convergence rate analysis provides a valuable mechanism to compare and improve of minimization algorithms. But there is a deeper significance with boosting: a convergence rate of $\mathcal{O}(\ln(1/\epsilon))$ means that, with a combination of just $\mathcal{O}(\ln(1/\epsilon))$ predictors, one can construct an $\epsilon$-optimal classifier, which is crucial to both the computational efficiency and statistical stability of this predictor.

The contribution of this manuscript is to provide a tight convergence theory for a large class of losses, including the exponential and logistic losses, which has heretofore resisted analysis. The goal is a general analysis without any assumptions (attainability of the minimum, or weak learnability),

however this manuscript also demonstrates how the classically understood scenarios of attainability and weak learnability can be understood directly from the sample and the weak learning class.

The organization is as follows. Section 2 provides a few pieces of background: how to encode the weak learning class and sample as a matrix, boosting as coordinate descent, and the primal objective function. Section 3 then gives the dual problem, max entropy. Given these tools, section 4 shows how to adjust the weak learning rate to a quantity which is useful without any assumptions. The first step towards convergence rates is then taken in section 5, which demonstrates that the weak learning rate is in fact a mechanism to convert between the primal and dual problems.

The convergence rates then follow: section 6 and section 7 discuss, respectively, the conditions under which classical weak learnability and (disjointly) attainability hold, both yielding the rate $\mathcal{O}(\ln(1/\epsilon))$, and finally section 8 shows how the general case may be decomposed into these two, and the conflicting optimization behavior leads to a degraded rate of $\mathcal{O}(1/\epsilon)$. The last section will also exhibit an $\Omega(1/\epsilon)$ lower bound for the logistic loss.

## 1.1 Related Work

The development of general convergence rates has a number of important milestones in the past decade. The first convergence result, albeit without any rates, is due to Collins et al. [8]; the work considered the improvement due to a single step, and as its update rule was less aggressive than the line search of boosting, it appears to imply general convergence. Next, Bickel et al. [6] showed a rate of $\mathcal{O}(\exp(1/\epsilon^2))$, where the assumptions of bounded second derivatives on compact sets are also necessary here.

Many extremely important cases have also been handled. The first is the original rate of $\mathcal{O}(\ln(1/\epsilon))$ for the exponential loss under the weak learning assumption [3]. Next, Rätsch et al. [9] showed, for a class of losses similar to those considered here, a rate of $\mathcal{O}(\ln(1/\epsilon))$ when the loss minimizer is attainable. The current manuscript provides another mechanism to analyze this case (with the same rate), which is crucial to being able to produce a general analysis. And, very recently, parallel to this work, Mukherjee et al. [10] established the general convergence under the exponential loss, with a rate of $\Theta(1/\epsilon)$. The same matrix, due to Schapire [11], was used to show the lower bound there as for the logistic loss here; their upper bound proof also utilized a decomposition theorem.

It is interesting to mention that, for many variants of boosting, general convergence rates were known. Specifically, once it was revealed that boosting is trying to be not only correct but also have large margins [12], much work was invested into methods which explicitly maximized the margin [13], or penalized variants focused on the inseparable case [14, 15]. These methods generally impose some form of regularization [15], which grants attainability of the risk minimizer, and allows standard techniques to grant general convergence rates. Interestingly, the guarantees in those works cited in this paragraph are $\mathcal{O}(1/\epsilon^2)$.

## 2 Setup

A view of boosting, which pervades this manuscript, is that the action of the weak learning class upon the sample can be encoded as a matrix [9, 15]. Let a sample $\mathcal{S} := \{(x_i, y_i)\}_1^m \subseteq (\mathcal{X} \times \mathcal{Y})^m$ and a weak learning class $\mathcal{H}$ be given. For every $h \in \mathcal{H}$, let $\mathcal{S}|_h$ denote the projection onto $\mathcal{S}$ induced by $h$; that is, $\mathcal{S}|_h$ is a vector of length $m$, with coordinates $(\mathcal{S}|_h)_i = y_i h(x_i)$. If the set of all such columns $\{\mathcal{S}|_h : h \in \mathcal{H}\}$ is finite, collect them into the matrix $A \in \mathbb{R}^{m \times n}$. Let $a_i$ denote the $i^{\text{th}}$ row of $A$, corresponding to the example $(x_i, y_i)$, and let $\{h_j\}_1^n$ index the set of weak learners corresponding to columns of $A$. It is assumed, for convenience, that entries of $A$ are within $[-1, +1]$; relaxing this assumption merely scales the presented rates by a constant.

The setting considered in this manuscript is that this finite matrix can be constructed. Note that this can encode infinite classes, so long as they map to only $k < \infty$ values (in which case $A$ has at most $k^m$ columns). As another example, if the weak learners are binary, and $\mathcal{H}$ has VC dimension $d$, then Sauer's lemma grants that $A$ has at most $(m + 1)^d$ columns. This matrix view of boosting is thus similar to the interpretation of boosting performing descent on functional space, but the class complexity and finite sample have been used to reduce the function class to a finite object [16, 5].

---

**Routine** BOOST.
**Input** Convex function $f \circ A$.
**Output** Approximate primal optimum $\lambda$.

1. Initialize $\lambda_0 := \mathbf{0}_n$.
2. For $t = 1, 2, \ldots$, while $\nabla(f \circ A)(\lambda_{t-1}) \neq \mathbf{0}_n$:
   (a) Choose column $j_t := \operatorname{argmax}_j |\nabla(f \circ A)(\lambda_{t-1})^\top \mathbf{e}_j|$.
   (b) Line search: $\alpha_t$ apx. minimizes $\alpha \mapsto (f \circ A)(\lambda_{t-1} + \alpha \mathbf{e}_{j_t})$.
   (c) Update $\lambda_t := \lambda_{t-1} + \alpha_t \mathbf{e}_{j_t}$.
3. Return $\lambda_{t-1}$.

---

Figure 1: $l^1$ steepest descent [17, Algorithm 9.4] applied to $f \circ A$.

To make the connection to boosting, the missing ingredient is the loss function. Let $\mathbb{G}_0$ denote the set of loss functions $g$ satisfying: $g$ is twice continuously differentiable, $g'' > 0$ (which implies strict convexity), and $\lim_{x \to \infty} g(x) = 0$. (A few more conditions will be added in section 5 to prove convergence rates, but these properties suffice for the current exposition.) Crucially, the exponential loss $\exp(-x)$ from AdaBoost and the logistic loss $\ln(1 + \exp(-x))$ are in $\mathbb{G}_0$ (and the eventual $\mathbb{G}$).

Boosting determines some weighting $\lambda \in \mathbb{R}^n$ of the columns of $A$, which correspond to weak learners in $\mathcal{H}$. The (unnormalized) margin of example $i$ is thus $\langle a_i, \lambda \rangle = \mathbf{e}_i^\top A \lambda$, where $\mathbf{e}_i$ is an indicator vector. Since the prediction on $x_i$ is $\mathbb{1}[\langle a_i, \lambda \rangle \geq 0]$, it follows that $A\lambda > \mathbf{0}_m$ (where $\mathbf{0}_m$ is the zero vector) implies a training error of zero. As such, boosting solves the minimization problem

$$\inf_{\lambda \in \mathbb{R}^n} \sum_{i=1}^m g(\langle a_i, \lambda \rangle) = \inf_{\lambda \in \mathbb{R}^n} \sum_{i=1}^m g(\mathbf{e}_i^\top A \lambda) = \inf_{\lambda \in \mathbb{R}^n} f(A\lambda) = \inf_{\lambda \in \mathbb{R}^n} (f \circ A)(\lambda) =: \bar{f}_A, \quad (2.1)$$

where $f : \mathbb{R}^m \to \mathbb{R}$ is the convenience function $f(x) = \sum_i g((x)_i)$, and in the present problem denotes the (unnormalized) empirical risk. $\bar{f}_A$ will denote the optimal objective value.

The infimum in eq. (2.1) may well not be attainable. Suppose there exists $\lambda'$ such that $A\lambda' > \mathbf{0}_m$ (theorem 6.1 will show that this is equivalent to the weak learning assumption). Then

$$0 \leq \inf_{\lambda \in \mathbb{R}^n} f(A\lambda) \leq \inf \{f(A\lambda) : \lambda = c\lambda', c > 0\} = \inf_{c > 0} f(c(A\lambda')) = 0.$$

On the other hand, for any $\lambda \in \mathbb{R}^n$, $f(A\lambda) > 0$. Thus the infimum is never attainable when weak learnability holds.

The template boosting algorithm appears in fig. 1, formulated in terms of $f \circ A$ to make the connection to coordinate descent as clear as possible. To interpret the gradient terms, note that

$$(\nabla(f \circ A)(\lambda))_j = (A^\top \nabla f(A\lambda))_j = \sum_{i=1}^m g'(\langle a_i, \lambda \rangle) h_j(x_i) y_i,$$

which is the expected correlation of $h_j$ with the target labels according to an unnormalized distribution with weights $g'(\langle a_i, \lambda \rangle)$. The stopping condition $\nabla(f \circ A)(\lambda) = \mathbf{0}_m$ means: either the distribution is degenerate (it is exactly zero), or every weak learner is uncorrelated with the target.

As such, eq. (2.1) represents an equivalent formulation of boosting, with one minor modification: the column (weak learner) selection has an absolute value. But note that this is the same as closing $\mathcal{H}$ under complementation (i.e., for any $h \in \mathcal{H}$, there exists $h'$ with $h(x) = -h'(x)$), which is assumed in many theoretical treatments of boosting.

In the case of the exponential loss with binary weak learners, the line search step has a convenient closed form; but for other losses, or even for the exponential loss but with confidence-rated predictors, there may not be a closed form. Moreover, this univariate search problem may lack a minimizer. To produce the eventual convergence rates, this manuscript utilizes a step size minimizing an upper bounding quadratic (which is guaranteed to exist); if instead a standard iterative line search guarantee were used, rates would only degrade by a constant factor [17, section 9.3.1].

As a final remark, consider the rows $\{a_i\}_1^m$ of $A$ as a collection of $m$ points in $\mathbb{R}^n$. Due to the form of $g$, BOOST is therefore searching for a halfspace, parameterized by a vector $\lambda$, which contains all of the points. Sometimes such a halfspace may not exist, and $g$ applies a smoothly increasing penalty to points that are farther and farther outside it.

## 3  Dual Problem

This section provides the convex dual to eq. (2.1). The relevance of the dual to convergence rates is as follows. First, although the primal optimum may not be attainable, the dual optimum is always attainable—this suggests a strategy of mapping the convergence strategy to the dual, where there exists a clear notion of progress to the optimum. Second, this section determines the dual feasible set—the space of dual variables or what the boosting literature typically calls *unnormalized weights*. Understanding this set is key to relating weak learnability, attainability, and general instances.

Before proceeding, note that the dual formulation will make use of the Fenchel conjugate $h^*(\phi) = \sup_{x \in \mathrm{dom}(h)} \langle x, \phi \rangle - h(x)$, a concept taking a central place in convex analysis [18, 19]. Interestingly, the Fenchel conjugates to the exponential and logistic losses are respectively the Boltzmann-Shannon and Fermi-Dirac entropies [19, Commentary, section 3.3], and thus the dual is explicitly performing entropy maximization (cf. lemma C.2). As a final piece of notation, denote the kernel of a matrix $B \in \mathbb{R}^{m \times n}$ by $\mathrm{Ker}(B) = \{\phi \in \mathbb{R}^n : B\phi = \mathbf{0}_m\}$.

**Theorem 3.1.** *For any $A \in \mathbb{R}^{m \times n}$ and $g \in \mathbb{G}_0$ with $f(x) = \sum_i g((x)_i)$,*

$$\inf \{f(A\lambda) : \lambda \in \mathbb{R}^n\} = \sup \{-f^*(-\phi) : \phi \in \Phi_A\}, \tag{3.2}$$

*where $\Phi_A := \mathrm{Ker}(A^\top) \cap \mathbb{R}_+^m$ is the dual feasible set. The dual optimum $\psi_A$ is unique and attainable. Lastly, $f^*(\phi) = \sum_{i=1}^m g^*((\phi)_i)$.*

The dual feasible set $\Phi_A = \mathrm{Ker}(A^\top) \cap \mathbb{R}_+^m$ has a strong interpretation. Suppose $\psi \in \Phi_A$; then $\psi$ is a nonnegative vector (since $\psi \in \mathbb{R}_+^m$), and, for any $j$, $0 = (\phi^\top A)_j = \sum_{i=1}^m \phi_i y_i h_j(x_i)$. That is to say, every nonzero feasible dual vector provides a (an unnormalized) distribution upon which every weak learner is uncorrelated! Furthermore, recall that the weak learning assumption states that under any weighting of the input, there exists a correlated weak learner; as such, weak learnability necessitates that the dual feasible set contains only the zero vector.

There is also a geometric interpretation. Ignoring the constraint, $-f^*$ attains its maximum at some rescaling of the uniform distribution (for details, please see lemma C.2). As such, the constrained dual problem is aiming to write the origin as a high entropy convex combination of the points $\{a_i\}_1^m$.

## 4  A Generalized Weak Learning Rate

The weak learning rate was critical to the original convergence analysis of AdaBoost, providing a handle on the progress of the algorithm. Recall that the quantity appeared in the denominator of the convergence rate, and a *weak learning assumption* critically provided that this quantity is nonzero. This section will generalize the weak learning rate to a quantity which is always positive, without any assumptions.

Note briefly that this manuscript will differ slightly from the norm in that weak learning will be a purely *sample-specific* concept. That is, the concern here is convergence, and all that matters is the sample $\mathcal{S} = \{(x_i, y_i)\}_1^m$, as encoded in $A$; it doesn't matter if there are wild points outside this sample, because the algorithm has no access to them.

This distinction has the following implication. The usual weak learning assumption states that there exists no uncorrelating distribution over the input *space*. This of course implies that any training sample $\mathcal{S}$ used by the algorithm will also have this property; however, it suffices that there is no distribution over the input *sample* $\mathcal{S}$ which uncorrelates the weak learners from the target.

Returning to task, the weak learning assumption posits the existence of a constant, the weak learning rate $\gamma$, which lower bounds the correlation of the best weak learner with the target for any distribu-

tion. Stated in terms of the matrix $A$,

$$0 < \gamma = \inf_{\substack{\phi \in \mathbb{R}^m_+ \\ \|\phi\|=1}} \max_{j \in [n]} \left| \sum_{i=1}^{m} (\phi)_i y_i h_j(x_i) \right| = \inf_{\phi \in \mathbb{R}^m_+ \setminus \{\mathbf{0}_m\}} \frac{\|A^\top \phi\|_\infty}{\|\phi\|_1} = \inf_{\phi \in \mathbb{R}^m_+ \setminus \{\mathbf{0}_m\}} \frac{\|A^\top \phi\|_\infty}{\|\phi - \mathbf{0}_m\|_1}.$$

(4.1)

The only way this quantity can be positive is if $\phi \notin \mathrm{Ker}(A^\top) \cap \mathbb{R}^m_+ = \Phi_A$, meaning the dual feasible set is exactly $\{\mathbf{0}_m\}$. As such, one candidate adjustment is to simply replace $\{\mathbf{0}_m\}$ with the dual feasible set:

$$\gamma' := \inf_{\phi \in \mathbb{R}^m_+ \setminus \Phi_A} \frac{\|A^\top \phi\|_\infty}{\inf_{\psi \in \Phi_A} \|\phi - \psi\|_1}.$$

Indeed, by the forthcoming proposition 4.3, $\gamma' > 0$ as desired. Due to technical considerations which will be postponed until the various convergence rates, it is necessary to tighten this definition with another set.

**Definition 4.2.** For a given matrix $A \in \mathbb{R}^{m \times n}$ and set $S \subseteq \mathbb{R}^m$, define

$$\gamma(A, S) := \inf \left\{ \frac{\|A^\top \phi\|_\infty}{\inf_{\psi \in S \cap \mathrm{Ker}(A^\top)} \|\phi - \psi\|_1} : \phi \in S \setminus \mathrm{Ker}(A^\top) \right\}. \qquad \Diamond$$

Crucially, for the choices of $S$ pertinent here, this quantity is always positive.

**Proposition 4.3.** *Let $A \neq \mathbf{0}_{m \times n}$ and polyhedron $S$ be given. If $S \cap \mathrm{Ker}(A^\top) \neq \emptyset$ and $S$ has nonempty interior, $\gamma(A, S) \in (0, \infty)$.*

To simplify discussion, the following projection and distance notation will be used in the sequel:

$$\mathsf{P}_C^p(x) \in \underset{y \in C}{\mathrm{Argmin}} \|y - x\|_p, \qquad \qquad \mathsf{D}_C^p(x) = \|x - \mathsf{P}_C^p(x)\|_p,$$

with some arbitrary choice made when the minimizer is not unique.

# 5 Prelude to Convergence Rates: Three Alternatives

The pieces are in place to finally sketch how the convergence rates may be proved. This section identifies how the weak learning rate $\gamma(A, S)$ can be used to convert the standard gradient guarantees into something which can be used in the presence of no attainable minimum. To close, three basic optimization scenarios are identified, which lead to the following three sections on convergence rates. But first, it is a good time to define the final loss function class.

**Definition 5.1.** Every $g \in \mathbb{G}$ satisfies the following properties. First, $g \in \mathbb{G}_0$. Next, for any $x \in \mathbb{R}^m$ satisfying $f(x) \leq f(A\lambda_0)$, and for any coordinate $(x)_i$, there exist constants $\eta > 0$ and $\beta > 0$ such that $g''((x)_i) \leq \eta g((x)_i)$ and $g((x)_i) \leq -\beta g'((x)_i)$. $\qquad \Diamond$

The exponential loss is in this class with $\eta = \beta = 1$ since $\exp(\cdot)$ is a fixed point with respect to the differentiation operator. Furthermore, as is verified in remark F.1 of the full version, the logistic loss is also in this class, with $\eta = 2^m/(m \ln(2))$ and $\beta \leq 1 + 2^m$. Intuitively, $\eta$ and $\beta$ encode how similar some $g \in \mathbb{G}$ is to the exponential loss, and thus these parameters can degrade radically. However, outside the weak learnability case, the other terms in the bounds here will also incur a penalty of the form $e^m$ for the exponential loss, and there is some evidence that this is unavoidable (see the lower bounds in Mukherjee et al. [10] or the upper bounds in Rätsch et al. [9]).

Next, note how the standard guarantee for coordinate descent methods can lead to guarantees on the progress of the algorithm in terms of dual distances, thanks to $\gamma(A, S)$.

**Proposition 5.2.** *For any $t$, $A \neq \mathbf{0}^{m \times n}$, $S \supseteq \{-\nabla f(A\lambda_t)\}$ with $\gamma(A, S) > 0$, and $g \in \mathbb{G}$,*

$$f(A\lambda_{t+1}) - \bar{f}_A \leq f(A\lambda_t) - \bar{f}_A - \frac{\gamma(A, S)^2 \mathsf{D}_{S \cap \mathrm{Ker}(A^\top)}^1(-\nabla f(A\lambda_t))^2}{2\eta f(A\lambda_t)}.$$

*Proof.* The stopping condition grants $-\nabla f(A\lambda_t) \notin \mathrm{Ker}(A^\top)$. Thus, by definition of $\gamma(A, S)$,

$$\gamma(A, S) = \inf_{\phi \in S \setminus \mathrm{Ker}(A^\top)} \frac{\|A^\top \phi\|_\infty}{\mathsf{D}_{S \cap \mathrm{Ker}(A^\top)}^1(\phi)} \leq \frac{\|A^\top \nabla f(A\lambda_t)\|_\infty}{\mathsf{D}_{S \cap \mathrm{Ker}(A^\top)}^1(-\nabla f(A\lambda_t))}.$$

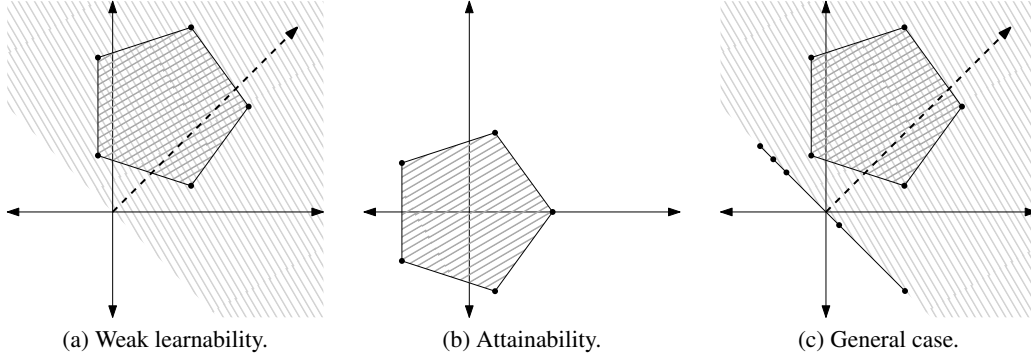

|  (a) Weak learnability. | (b) Attainability. | (c) General case. |

Figure 2: Viewing the rows $\{a_i\}_1^m$ of $A$ as points in $\mathbb{R}^n$, boosting seeks a homogeneous halfspace, parameterized by a normal $\lambda \in \mathbb{R}^n$, which contains all $m$ points. The dual, on the other hand, aims to express the origin as a high entropy convex combination of the rows. The convergence rate and dynamics of this process are controlled by $A$, which dictates one of the three above scenarios.

Combined with a standard guarantee of coordinate descent progress (cf. lemma F.2),

$$f(A\lambda_t) - f(A\lambda_{t+1}) \geq \frac{\|A^\top \nabla f(A\lambda_t)\|_\infty^2}{2\eta f(A\lambda_t)} \geq \frac{\gamma(A,S)^2 \mathsf{D}_{S \cap \mathrm{Ker}(A^\top)}^1 (-\nabla f(A\lambda_t))^2}{2\eta f(A\lambda_t)}.$$

Subtracting $\bar{f}_A$ from both sides and rearranging yields the statement. $\qquad\square$

Recall the interpretation of boosting closing section 2: boosting seeks a halfspace, parameterized by $\lambda \in \mathbb{R}^n$, which contains the points $\{a_i\}_1^m$. Progress onward from proposition 5.2 will be divided into three cases, each distinguished by the kind of halfspace which boosting can reach.

These cases appear in fig. 2. The first case is weak learnability: positive margins can be attained on each example, meaning a halfspace exists which strictly contains all points. Boosting races to push all these margins unboundedly large, and has a convergence rate $\mathcal{O}(\ln(1/\epsilon))$. Next is the case that no halfspace contains the points within its interior: either any such halfspace has the points on its boundary, or no such halfspace exists at all (the degenerate choice $\lambda = \mathbf{0}_n$). This is the case of attainability: boosting races towards finite margins at the rate $\mathcal{O}(\ln(1/\epsilon))$.

The final situation is a mix of the two: there exists a halfspace with some points on the boundary, some within its interior. Boosting will try to push some margins to infinity, and keep others finite. These two desires are at odds, and the rate degrades to $\mathcal{O}(1/\epsilon)$. Less metaphorically, the analysis will proceed by decomposing this case into the previous two, applying the above analysis in parallel, and then stitching the result back together. It is precisely while stitching up that an incompatibility arises, and the rate degrades. This is no artifact: a lower bound will be shown for the logistic loss.

## 6 Convergence Rate under Weak Learnability

To start this section, the following result characterizes weak learnability, including the earlier relationship to the dual feasible set (specifically, that it is precisely the origin), and, as analyzed by many authors, the relationship to separability [1, 9, 15].

**Theorem 6.1.** *For any $A \in \mathbb{R}^{m \times n}$ and $g \in \mathbb{G}$ the following conditions are equivalent:*

$$\exists \lambda \in \mathbb{R}^n \centerdot A\lambda \in \mathbb{R}_{++}^m, \tag{6.2}$$

$$\inf_{\lambda \in \mathbb{R}^n} f(A\lambda) = 0, \tag{6.3}$$

$$\psi_A = \mathbf{0}_m, \tag{6.4}$$

$$\Phi_A = \{\mathbf{0}_m\}. \tag{6.5}$$

The equivalence means the presence of any of these properties suffices to indicate weak learnability. The last two statements encode the usual distributional version of the weak learning assumption.

The first encodes the fact that there exists a homogeneous halfspace containing all points within its interior; this encodes separability, since removing the factor $y_i$ from the definition of $a_i$ will place all negative points outside the halfspace. Lastly, the second statement encodes the fact that the empirical risk approaches zero.

**Theorem 6.6.** *Suppose $A\lambda > \mathbf{0}_m$ and $g \in \mathbb{G}$; then $\gamma(A, \mathbb{R}_+^m) > 0$, and for all $t$,*

$$f(A\lambda_t) - \bar{f}_A \leq f(A\lambda_0)\left(1 - \frac{\gamma(A, \mathbb{R}_+^m)^2}{2\beta^2\eta}\right)^t.$$

*Proof.* By theorem 6.1, $\mathbb{R}_+^m \cap \mathrm{Ker}(A^\top) = \Phi_A = \{\mathbf{0}_m\}$, which combined with $g \leq -\beta g'$ gives

$$\mathrm{D}_{\Phi_A}^1(-\nabla f(A\lambda_t)) = \inf_{\psi \in \Phi_A} \| -\nabla f(A\lambda_t) - \psi\|_1 = \|\nabla f(A\lambda_t)\|_1 \geq f(A\lambda_t)/\beta.$$

Plugging this and $\bar{f}_A = 0$ (again by theorem 6.1) along with polyhedron $\mathbb{R}_+^m \supseteq -\nabla f(\mathbb{R}^m)$ (whereby $\gamma(A, \mathbb{R}_+^m) > 0$ by proposition 4.3 since $\psi_A \in \mathbb{R}_+^m$) into proposition 5.2 gives

$$f(A\lambda_{t+1}) \leq f(A\lambda_t) - \frac{\gamma(A, \mathbb{R}_+^m)^2 f(A\lambda_t)}{2\beta^2\eta} = f(A\lambda_t)\left(1 - \frac{\gamma(A, \mathbb{R}_+^m)^2}{2\beta^2\eta}\right),$$

and recursively applying this inequality yields the result. $\qquad\square$

Since the present setting is weak learnability, note by (4.1) that the choice of polyhedron $\mathbb{R}_+^m$ grants that $\gamma(A, \mathbb{R}_+^m)$ is exactly the original weak learning rate. When specialized for the exponential loss (where $\eta = \beta = 1$), the bound becomes $(1 - \gamma(A, \mathbb{R}_+^m)^2/2)^t$, which exactly recovers the bound of Schapire and Singer [20], although via different analysis.

In general, solving for $t$ in the expression $\epsilon = \frac{f(A\lambda_t) - \bar{f}_A}{f(A\lambda_0) - \bar{f}_A} \leq \left(1 - \frac{\gamma(f,A)^2}{2\beta^2\eta}\right)^t \leq \exp\left(-\frac{\gamma(f,A)^2 t}{2\beta^2\eta}\right)$ reveals that $t \leq \frac{2\beta^2\eta}{\gamma(A,S)^2}\ln(1/\epsilon)$ iterations suffice to reach error $\epsilon$. Recall that $\beta$ and $\eta$, in the case of the logistic loss, have only been bounded by quantities like $2^m$. While it is unclear if this analysis of $\beta$ and $\eta$ was tight, note that it is plausible that the logistic loss is slower than the exponential loss in this scenario, as it works less in initial phases to correct minor margin violations.

## 7 Convergence Rate under Attainability

**Theorem 7.1.** *For any $A \in \mathbb{R}^{m \times n}$ and $g \in \mathbb{G}$, the following conditions are equivalent:*

$$\forall \lambda \in \mathbb{R}^n \,\boldsymbol{.}\, A\lambda \notin \mathbb{R}_+^m \setminus \{\mathbf{0}_m\}, \tag{7.2}$$

$$f \circ A \text{ has minimizers}, \tag{7.3}$$

$$\psi_A \in \mathbb{R}_{++}^m, \tag{7.4}$$

$$\Phi_A \cap \mathbb{R}_{++}^m \neq \emptyset. \tag{7.5}$$

Interestingly, as revealed in (7.4) and (7.5), attainability entails that the dual has fully interior points, and furthermore that the dual optimum is interior. On the other hand, under weak learnability, eq. (6.4) provided that the dual optimum has zeros at every coordinate. As will be made clear in section 8, the primal and dual weights have the following dichotomy: either the margin $\langle a_i, \lambda \rangle$ goes to infinity and $(\psi_A)_i$ goes to zero, or the margin stays finite and $(\psi_A)_i$ goes to some positive value.

**Theorem 7.6.** *Suppose $A \neq \mathbf{0}_{m \times n}$, $g \in \mathbb{G}$, and the infimum of eq. (2.1) is attainable. Then there exists a (compact) tightest axis-aligned retangle $\mathcal{C}$ containing the initial level set, and $f$ is strongly convex with modulus $c > 0$ over $\mathcal{C}$. Finally, $\gamma(A, -\nabla f(\mathcal{C})) > 0$, and for all $t$,*

$$f(A\lambda_t) - \bar{f}_A \leq (f(\mathbf{0}_m) - \bar{f}_A)\left(1 - \frac{c\gamma(A, -\nabla f(\mathcal{C}))^2}{\eta f(A\lambda_0)}\right)^t.$$

In other words, $t \leq \frac{\eta f(A\lambda_0)}{c\gamma(A, -\nabla f(\mathcal{C}))^2}\ln(\frac{1}{\epsilon})$ iterations suffice to reach error $\epsilon$. The appearance of a modulus of strong convexity $c$ (i.e., a lower bound on the eigenvalues of the Hessian of $f$) may seem surprising, and sketching the proof illuminates its appearance and subsequent function.

When the infimum is attainable, every margin $\langle a_i, \lambda \rangle$ converges to some finite value. In fact, they all remain bounded: (7.2) provides that no halfspace contains all points, so if one margin becomes positive and large, another becomes negative and large, giving a terrible objective value. But objective values never increase with coordinate descent. To finish the proof, strong convexity (i.e., quadratic lower bounds in the primal) grants quadratic upper bounds in the dual, which can be used to bound the dual distance in proposition 5.2, and yield the desired convergence rate. This approach fails under weak learnability—some primal weights grow unboundedly, all dual weights shrink to zero, and no compact set contains all margins.

# 8 General Convergence Rate

The final characterization encodes two principles: the rows of $A$ may be partitioned into two matrices $A_0, A_+$ which respectively satisfy theorem 6.1 and theorem 7.1, and that these two subproblems affect the optimization problem essentially independently.

**Theorem 8.1.** *Let $A_0 \in \mathbb{R}^{z \times n}$, $A_+ \in \mathbb{R}^{p \times n}$, and $g \in \mathbb{G}$ be given. Set $m := z + p$, and $A \in \mathbb{R}^{m \times n}$ to be the matrix obtained by stacking $A_0$ on top of $A_+$. The following conditions are equivalent:*

$$(\exists \lambda \in \mathbb{R}^n \bullet A_0 \lambda \in \mathbb{R}_{++}^z \wedge A_+ \lambda = \mathbf{0}_p) \wedge (\forall \lambda \in \mathbb{R}^n \bullet A_+ \lambda \notin \mathbb{R}_+^p \setminus \{\mathbf{0}_p\}), \tag{8.2}$$

$$( \inf_{\lambda \in \mathbb{R}^n} f(A\lambda) = \inf_{\lambda \in \mathbb{R}^n} f(A_+\lambda)) \wedge ( \inf_{\lambda \in \mathbb{R}^n} f(A_0\lambda) = 0) \wedge f \circ A_+ \text{ has minimizers}, \tag{8.3}$$

$$\psi_A = \begin{bmatrix} \psi_{A_0} \\ \psi_{A_+} \end{bmatrix} \text{ with } \psi_{A_0} = \mathbf{0}_z \wedge \psi_{A_+} \in \mathbb{R}_{++}^p, \tag{8.4}$$

$$(\Phi_{A_0} = \{\mathbf{0}_z\}) \wedge (\Phi_{A_+} \cap \mathbb{R}_{++}^p \neq \emptyset) \wedge (\Phi_A = \Phi_{A_0} \times \Phi_{A_+}). \tag{8.5}$$

To see that any matrix $A$ falls into one of the three scenarios here, fix a loss function $g$, and recall from theorem 3.1 that $\psi_A$ is unique. In particular, the set of zero entries in $\psi_A$ exactly specifies which of the three scenarios hold, the current scenario allowing for simultaneous positive and zero entries. Although this reasoning made use of $\psi_A$, note that it is $A$ which dictates the behavior: in fact, as is shown in remark I.1 of the full version, the decomposition is unique.

Returning to theorem 8.1, the geometry of fig. 2c is provided by (8.2) and (8.5). The analysis will start from (8.3), which allows the primal problem to be split into two pieces, which are then individually handled precisely as in the preceding sections. To finish, (8.5) will allow these pieces to be stitched together.

**Theorem 8.6.** *Suppose $A \neq \mathbf{0}_{m \times n}$, $g \in \mathbb{G}$, $\psi_A \in \mathbb{R}_+^m \setminus \mathbb{R}_{++}^m \setminus \{\mathbf{0}_m\}$, and the notation from theorem 8.1. Set $w := \sup_t \|\nabla f(A_+\lambda_t) + \mathsf{P}_{\Phi_{A_+}}^1 (-\nabla f(A_+\lambda_t))\|_1$. Then $w < \infty$, and there exists a tightest cube $\mathcal{C}_+$ so that $\mathcal{C}_+ \supseteq \{x \in \mathbb{R}^p : f(x) \leq f(A\lambda_0)\}$, and let $c > 0$ be the modulus of strong convexity of $f$ over $\mathcal{C}_+$. Then $\gamma(A, \mathbb{R}_+^z \times -\nabla f(\mathcal{C}_+)) > 0$, and for all $t$, $f(A\lambda_t) - \bar{f}_A \leq 2f(A\lambda_0)/ \left( (t+1) \min \left\{ 1, \gamma(A, \mathbb{R}_+^z \times -\nabla f(\mathcal{C}_+))^2/((\beta + w/(2c))^2\eta) \right\} \right)$.*

(In the case of the logistic loss, $w \leq \sup_{x \in \mathbb{R}^m} \|\nabla f(x)\|_1 \leq m$.)

As discussed previously, the bounds deteriorate to $\mathcal{O}(1/\epsilon)$ because the finite and infinite margins sought by the two pieces $A_0, A_+$ are in conflict. For a beautifully simple, concrete case of this, consider the following matrix, due to Schapire [11]:

$$S := \begin{bmatrix} -1 & +1 \\ +1 & -1 \\ +1 & +1 \end{bmatrix}.$$

The optimal solution here is to push both coordinates of $\lambda$ unboundedly positive, with margins approaching $(0, 0, \infty)$. But pushing any coordinate $\lambda_i$ too quickly will increase the objective value, rather than decreasing it. In fact, this instance will provide a lower bound, and the mechanism of the proof shows that the primal weights grow extremely slowly, as $\mathcal{O}(\ln(t))$.

**Theorem 8.7.** *Using the logistic loss and exact line search, for any $t \geq 1$, $f(S\lambda_t) - \bar{f}_S \geq 1/(8t)$.*

**Acknowledgement**

The author thanks Sanjoy Dasgupta, Daniel Hsu, Indraneel Mukherjee, and Robert Schapire for valuable conversations. The NSF supported this work under grants IIS-0713540 and IIS-0812598.

# References

[1] Robert E. Schapire. The strength of weak learnability. *Machine Learning*, 5:197–227, July 1990.

[2] Yoav Freund. Boosting a weak learning algorithm by majority. *Information and Computation*, 121(2):256–285, 1995.

[3] Yoav Freund and Robert E. Schapire. A decision-theoretic generalization of on-line learning and an application to boosting. *J. Comput. Syst. Sci.*, 55(1):119–139, 1997.

[4] Leo Breiman. Prediction games and arcing algorithms. *Neural Computation*, 11:1493–1517, October 1999.

[5] Jerome Friedman, Trevor Hastie, and Robert Tibshirani. Additive logistic regression: a statistical view of boosting. *Annals of Statistics*, 28, 1998.

[6] Peter J. Bickel, Yaacov Ritov, and Alon Zakai. Some theory for generalized boosting algorithms. *Journal of Machine Learning Research*, 7:705–732, 2006.

[7] Z. Q. Luo and P. Tseng. On the convergence of the coordinate descent method for convex differentiable minimization. *Journal of Optimization Theory and Applications*, 72:7–35, 1992.

[8] Michael Collins, Robert E. Schapire, and Yoram Singer. Logistic regression, AdaBoost and Bregman distances. *Machine Learning*, 48(1-3):253–285, 2002.

[9] Gunnar Rätsch, Sebastian Mika, and Manfred K. Warmuth. On the convergence of leveraging. In *NIPS*, pages 487–494, 2001.

[10] Indraneel Mukherjee, Cynthia Rudin, and Robert Schapire. The convergence rate of AdaBoost. In *COLT*, 2011.

[11] Robert E. Schapire. The convergence rate of AdaBoost. In *COLT*, 2010.

[12] Robert E. Schapire, Yoav Freund, Peter Barlett, and Wee Sun Lee. Boosting the margin: A new explanation for the effectiveness of voting methods. In *ICML*, pages 322–330, 1997.

[13] Gunnar Rätsch and Manfred K. Warmuth. Maximizing the margin with boosting. In *COLT*, pages 334–350, 2002.

[14] Manfred K. Warmuth, Karen A. Glocer, and Gunnar Rätsch. Boosting algorithms for maximizing the soft margin. In *NIPS*, 2007.

[15] Shai Shalev-Shwartz and Yoram Singer. On the equivalence of weak learnability and linear separability: New relaxations and efficient boosting algorithms. In *COLT*, pages 311–322, 2008.

[16] Llew Mason, Jonathan Baxter, Peter L. Bartlett, and Marcus R. Frean. Functional gradient techniques for combining hypotheses. In A.J. Smola, P.L. Bartlett, B. Schölkopf, and D. Schuurmans, editors, *Advances in Large Margin Classifiers*, pages 221–246, Cambridge, MA, 2000. MIT Press.

[17] Stephen P. Boyd and Lieven Vandenberghe. *Convex Optimization*. Cambridge University Press, 2004.

[18] Jean-Baptiste Hiriart-Urruty and Claude Lemaréchal. *Fundamentals of Convex Analysis*. Springer Publishing Company, Incorporated, 2001.

[19] Jonathan Borwein and Adrian Lewis. *Convex Analysis and Nonlinear Optimization*. Springer Publishing Company, Incorporated, 2000.

[20] Robert E. Schapire and Yoram Singer. Improved boosting algorithms using confidence-rated predictions. *Machine Learning*, 37(3):297–336, 1999.

[21] George B. Dantzig and Mukund N. Thapa. *Linear Programming 2: Theory and Extensions*. Springer, 2003.

[22] Adi Ben-Israel. Motzkin's transposition theorem, and the related theorems of Farkas, Gordan and Stiemke. In M. Hazewinkel, editor, *Encyclopaedia of Mathematics, Supplement III*. 2002.

